# Isotropic Hashing

**Weihao Kong, Wu-Jun Li**
Shanghai Key Laboratory of Scalable Computing and Systems
Department of Computer Science and Engineering, Shanghai Jiao Tong University, China
{kongweihao,liwujun}@cs.sjtu.edu.cn

## Abstract

Most existing hashing methods adopt some projection functions to project the o-riginal data into several dimensions of real values, and then each of these projected dimensions is quantized into one bit (zero or one) by thresholding. Typically, the variances of different projected dimensions are different for existing projection functions such as principal component analysis (PCA). Using the same number of bits for different projected dimensions is unreasonable because larger-variance dimensions will carry more information. Although this viewpoint has been widely accepted by many researchers, it is still not verified by either theory or experiment because no methods have been proposed to find a projection with equal variances for different dimensions. In this paper, we propose a novel method, called isotrop-ic hashing (IsoHash), to learn projection functions which can produce projected dimensions with isotropic variances (equal variances). Experimental results on real data sets show that IsoHash can outperform its counterpart with different vari-ances for different dimensions, which verifies the viewpoint that projections with isotropic variances will be better than those with anisotropic variances.

## 1 Introduction

Due to its fast query speed and low storage cost, hashing [1, 5] has been successfully used for approximate nearest neighbor (ANN) search [28]. The basic idea of hashing is to learn similarity-preserving binary codes for data representation. More specifically, each data point will be hashed into a compact binary string, and similar points in the original feature space should be hashed into close points in the hashcode space. Compared with the original feature representation, hashing has two advantages. One is the reduced storage cost, and the other is the constant or sub-linear query time complexity [28]. These two advantages make hashing become a promising choice for efficient ANN search in massive data sets [1, 5, 6, 9, 10, 14, 15, 17, 20, 21, 23, 26, 29, 30, 31, 32, 33, 34].

Most existing hashing methods adopt some projection functions to project the original data into several dimensions of real values, and then each of these projected dimensions is quantized into one bit (zero or one) by thresholding. Locality-sensitive hashing (LSH) [1, 5] and its extension-s [4, 18, 19, 22, 25] use simple random projections for hash functions. These methods are called *data-independent methods* because the projection functions are independent of training data. Anoth-er class of methods are called *data-dependent methods*, whose projection functions are learned from training data. Representative data-dependent methods include spectral hashing (SH) [31], anchor graph hashing (AGH) [21], sequential projection learning (SPL) [29], principal component analy-sis [13] based hashing (PCAH) [7], and iterative quantization (ITQ) [7, 8]. SH learns the hashing functions based on spectral graph partitioning. AGH adopts anchor graphs to speed up the com-putation of graph Laplacian eigenvectors, based on which the Nyström method is used to compute projection functions. SPL leans the projection functions in a sequential way that each function is designed to correct the errors caused by the previous one. PCAH adopts principal component anal-ysis (PCA) to learn the projection functions. ITQ tries to learn an orthogonal rotation matrix to refine the initial projection matrix learned by PCA so that the quantization error of mapping the data

to the vertices of binary hypercube is minimized. Compared to the data-dependent methods, the data-independent methods need longer codes to achieve satisfactory performance [7].

For most existing projection functions such as those mentioned above, the variances of different projected dimensions are different. Many researchers [7, 12, 21] have argued that using the same number of bits for different projected dimensions with unequal variances is unreasonable because larger-variance dimensions will carry more information. Some methods [7, 12] use orthogonal transformation to the PCA-projected data with the expectation of balancing the variances of different PCA dimensions, and achieve better performance than the original PCA based hashing. However, to the best of our knowledge, there exist no methods which can guarantee to learn a projection with equal variances for different dimensions. Hence, the viewpoint that using the same number of bits for different projected dimensions is unreasonable has still not been verified by either theory or experiment.

In this paper, a novel hashing method, called isotropic hashing (IsoHash), is proposed to learn a projection function which can produce projected dimensions with isotropic variances (equal variances). To the best of our knowledge, this is the first work which can learn projections with isotropic variances for hashing. Experimental results on real data sets show that IsoHash can outperform its counterpart with anisotropic variances for different dimensions, which verifies the intuitive viewpoint that projections with isotropic variances will be better than those with anisotropic variances. Furthermore, the performance of IsoHash is also comparable, if not superior, to the state-of-the-art methods.

## 2 Isotropic Hashing

### 2.1 Problem Statement

Assume we are given $n$ data points $\{\mathbf{x}_1, \mathbf{x}_2, \cdots, \mathbf{x}_n\}$ with $\mathbf{x}_i \in \mathbb{R}^d$, which form the columns of the data matrix $X \in \mathbb{R}^{d \times n}$. Without loss of generality, in this paper the data are assumed to be zero centered which means $\sum_{i=1}^{n} \mathbf{x}_i = \mathbf{0}$. The basic idea of hashing is to map each point $\mathbf{x}_i$ into a binary string $\mathbf{y}_i \in \{0, 1\}^m$ with $m$ denoting the code size. Furthermore, close points in the original space $\mathbb{R}^d$ should be hashed into similar binary codes in the code space $\{0, 1\}^m$ to preserve the similarity structure in the original space. In general, we compute the binary code of $\mathbf{x}_i$ as $\mathbf{y}_i = [h_1(\mathbf{x}_i), h_2(\mathbf{x}_i), \cdots, h_m(\mathbf{x}_i)]^T$ with $m$ binary hash functions $\{h_k(\cdot)\}_{k=1}^{m}$.

Because it is NP hard to directly compute the best binary functions $h_k(\cdot)$ for a given data set [31], most hashing methods adopt a two-stage strategy to learn $h_k(\cdot)$. In the projection stage, $m$ real-valued *projection functions* $\{f_k(\mathbf{x})\}_{k=1}^{m}$ are learned and each function can generate one real value. Hence, we have $m$ *projected dimensions* each of which corresponds to one projection function. In the quantization stage, the real-values are quantized into a binary string by thresholding.

Currently, most methods use one bit to quantize each projected dimension. More specifically, $h_k(\mathbf{x}_i) = sgn(f_k(\mathbf{x}_i))$ where $sgn(x) = 1$ if $x \geq 0$ and 0 otherwise. The exceptions of the quantization methods only contain AGH [21], DBQ [14] and MH [15], which use two bits to quantize each dimension. In sum, all of these methods adopt the same number (either one or two) of bits for different projected dimensions. However, the variances of different projected dimensions are unequal, and larger-variance dimensions typically carry more information. Hence, using the same number of bits for different projected dimensions with unequal variances is unreasonable, which has also been argued by many researchers [7, 12, 21]. Unfortunately, there exist no methods which can learn projection functions with equal variances for different dimensions. In the following content of this section, we present a novel model to learn projections with isotropic variances.

### 2.2 Model Formulation

The idea of our IsoHash method is to learn an *orthogonal matrix* to rotate the PCA projection matrix.

To generate a code of $m$ bits, PCAH performs PCA on $X$, and then use the top $m$ eigenvectors of the covariance matrix $XX^T$ as columns of the projection matrix $W \in \mathbb{R}^{d \times m}$. Here, top $m$ eigenvectors are those corresponding to the $m$ largest eigenvalues $\{\lambda_k\}_{k=1}^{m}$, generally arranged with the non-

increasing order $\lambda_1 \geq \lambda_2 \geq \cdots \geq \lambda_m$. Hence, the projection functions of PCAH are defined as follows: $f_k(\mathbf{x}) = \mathbf{w}_k^T \mathbf{x}$, where $\mathbf{w}_k$ is the $k$th column of $W$.

Let $\lambda = [\lambda_1, \lambda_2, \cdots, \lambda_m]^T$ and $\Lambda = \text{diag}(\lambda)$, where $\text{diag}(\lambda)$ denotes the diagonal matrix whose diagonal entries are formed from the vector $\lambda$. It is easy to prove that $W^T X X^T W = \Lambda$. Hence, the variance of the values $\{f_k(\mathbf{x}_i)\}_{i=1}^n$ on the $k$th projected dimension, which corresponds to the $k$th row of $W^T X$, is $\lambda_k$. Obviously, the variances for different PCA dimensions are anisotropic.

To get isotropic projection functions, the idea of our IsoHash method is to learn an orthogonal matrix $Q \in \mathbb{R}^{m \times m}$ which makes $Q^T W^T X X^T W Q$ become a matrix with equal diagonal values, i.e., $[Q^T W^T X X^T W Q]_{11} = [Q^T W^T X X^T W Q]_{22} = \cdots = [Q^T W^T X X^T W Q]_{mm}$. Here, $A_{ii}$ denotes the $i$th diagonal entry of a square matrix $A$, and a matrix $Q$ is said to be *orthogonal* if $Q^T Q = \mathbf{I}$ where $\mathbf{I}$ is an identity matrix whose dimensionality depends on the context. The effect of the orthogonal matrix $Q$ is to rotate the coordinate axes while keeping the Euclidean distances between any two points unchanged. It is easy to prove that the new projection functions of IsoHash are $f_k(\mathbf{x}) = (WQ)_k^T \mathbf{x}$ which have the same (isotropic) variance. Here $(WQ)_k$ denotes the $k$th column of $WQ$.

If we use $\text{tr}(A)$ to denote the trace of a symmetric matrix $A$, we have the following Lemma 1.

**Lemma 1.** *If $Q^T Q = \mathbf{I}$, $\text{tr}(Q^T A Q) = \text{tr}(A)$.*

Based on Lemma 1, we have $\text{tr}(Q^T W^T X X^T W Q) = \text{tr}(W^T X X^T W) = \text{tr}(\Lambda) = \sum_{i=1}^m \lambda_i$ if $Q^T Q = \mathbf{I}$. Hence, to make $Q^T W^T X X^T W Q$ become a matrix with equal diagonal values, we should set this diagonal value $a = \frac{\sum_{i=1}^m \lambda_i}{m}$.

Let

$$\mathbf{a} = [a_1, a_2, \cdots, a_m] \text{ with } a_i = a = \frac{\sum_{i=1}^m \lambda_i}{m}, \tag{1}$$

and

$$\mathcal{T}(\mathbf{z}) = \{T \in \mathbb{R}^{m \times m} | \text{diag}(T) = \text{diag}(\mathbf{z})\},$$

where $\mathbf{z}$ is a vector of length $m$, $\text{diag}(T)$ is overloaded to denote a diagonal matrix with the same diagonal entries of matrix $T$.

Based on our motivation of IsoHash, we can define the problem of IsoHash as follows:

**Problem 1.** *The **problem of IsoHash** is to find an orthogonal matrix $Q$ making $Q^T W^T X X^T W Q \in \mathcal{T}(\mathbf{a})$, where $\mathbf{a}$ is defined in (1).*

Then, we have the following Theorem 1:

**Theorem 1.** *Assume $Q^T Q = \mathbf{I}$ and $T \in \mathcal{T}(\mathbf{a})$. If $Q^T \Lambda Q = T$, $Q$ will be a solution to the problem of IsoHash.*

*Proof.* Because $W^T X X^T W = \Lambda$, we have $Q^T \Lambda Q = Q^T [W^T X X^T W] Q$. It is obvious that $Q$ will be a solution to the problem of IsoHash. $\square$

As in [2], we define

$$\mathcal{M}(\Lambda) = \{Q^T \Lambda Q | Q \in \mathcal{O}(m)\}, \tag{2}$$

where $\mathcal{O}(m)$ is the set of all orthogonal matrices in $\mathbb{R}^{m \times m}$, i.e., $Q^T Q = \mathbf{I}$.

According to Theorem 1, the problem of IsoHash is equivalent to finding an orthogonal matrix $Q$ for the following equation [2]:

$$||T - Z||_F = 0, \tag{3}$$

where $T \in \mathcal{T}(\mathbf{a})$, $Z \in \mathcal{M}(\Lambda)$, $|| \cdot ||_F$ denotes the Frobenius norm. Please note that for ease of understanding, we use the same notations as those in [2].

In the following content, we will use the Schur-Horn lemma [11] to prove that we can always find a solution to problem (3).

**Lemma 2.** *[Schur-Horn Lemma] Let* $\mathbf{c} = \{c_i\} \in \mathbb{R}^m$ *and* $\mathbf{b} = \{b_i\} \in \mathbb{R}^m$ *be real vectors in non-increasing order respectively [1], i.e.,* $c_1 \geq c_2 \geq \cdots \geq c_m$, $b_1 \geq b_2 \geq \cdots \geq b_m$. *There exists a Hermitian matrix* $H$ *with eigenvalues* $\mathbf{c}$ *and diagonal values* $\mathbf{b}$ *if and only if*

$$\sum_{i=1}^{k} b_i \leq \sum_{i=1}^{k} c_i, \text{ for any } k = 1, 2, ..., m,$$

$$\sum_{i=1}^{m} b_i = \sum_{i=1}^{m} c_i.$$

*Proof.* Please refer to Horn's article [11]. □

Base on Lemma 2, we have the following Theorem 2.

**Theorem 2.** *There exists a solution to the IsoHash problem in (3). And this solution is in the intersection of* $\mathcal{T}(\mathbf{a})$ *and* $\mathcal{M}(\Lambda)$.

*Proof.* Because $\lambda_1 \geq \lambda_2 \geq \cdots \geq \lambda_m$ and $a_1 = a_2 = \cdots = a_m = \frac{\sum_{i=1}^{m} \lambda_i}{m}$, it is easy to prove that $\frac{\sum_{i=1}^{k} \lambda_i}{k} \geq \frac{\sum_{i=1}^{m} \lambda_i}{m}$ for any $k$. Hence, $\sum_{i=1}^{k} \lambda_i = k \times \frac{\sum_{i=1}^{k} \lambda_i}{k} \geq k \times \frac{\sum_{i=1}^{m} \lambda_i}{m} = \sum_{i=1}^{k} a_i$. Furthermore, we can prove that $\sum_{i=1}^{m} \lambda_i = \sum_{i=1}^{m} a_i$. According to Lemma 2, there exists a Hermitian matrix $H$ with eigenvalues $\lambda$ and diagonal values $\mathbf{a}$.
Moreover, we can prove that $H$ is in the intersection of $\mathcal{T}(\mathbf{a})$ and $\mathcal{M}(\Lambda)$, i.e., $H \in \mathcal{T}(\mathbf{a})$ and $H \in \mathcal{M}(\Lambda)$. □

According to Theorem 2, to find a $Q$ solving the problem in (3) is equivalent to finding the intersection point of $\mathcal{T}(\mathbf{a})$ and $\mathcal{M}(\Lambda)$, which is just an inverse eigenvalue problem called SHIEP in [2].

## 2.3 Learning

The problem in (3) can be reformulated as the following optimization problem:

$$\underset{Q:T\in\mathcal{T}(\mathbf{a}),Z\in\mathcal{M}(\Lambda)}{\operatorname{argmin}} ||T - Z||_F. \tag{4}$$

As in [2], we propose two algorithms to learn $Q$: one is called *lift and projection (LP)*, and the other is called *gradient flow (GF)*. For ease of understanding, we use the same notations as those in [2], and some proofs of theorems are omitted. The readers can refer to [2] for the details.

### 2.3.1 Lift and Projection

The main idea of lift and projection (LP) algorithm is to alternate between the following two steps:

- Lift step:
  Given a $T^{(k)} \in \mathcal{T}(\mathbf{a})$, we find the point $Z^{(k)} \in \mathcal{M}(\Lambda)$ such that $||T^{(k)} - Z^{(k)}||_F = dist(T^{(k)}, \mathcal{M}(\Lambda))$, where $dist(T^{(k)}, \mathcal{M}(\Lambda))$ denotes the minimum distance between $T^{(k)}$ and the points in $\mathcal{M}(\Lambda)$.

- Projection step:
  Given a $Z^{(k)}$, we find $T^{(k+1)} \in \mathcal{T}(\mathbf{a})$ such that $||T^{(k+1)} - Z^{(k)}||_F = dist(\mathcal{T}(\mathbf{a}), Z^{(k)})$, where $dist(\mathcal{T}(\mathbf{a}), Z^{(k)})$ denotes the minimum distance between $Z^{(k)}$ and the points in $\mathcal{T}(\mathbf{a})$.

We call $Z^{(k)}$ a lift of $T^{(k)}$ onto $\mathcal{M}(\Lambda)$ and $T^{(k+1)}$ a projection of $Z^{(k)}$ onto $\mathcal{T}(\mathbf{a})$. The projection operation is easy to complete. Suppose $Z^{(k)} = [z_{ij}]$, then $T^{(k+1)} = [t_{ij}]$ must be given by

$$t_{ij} = \begin{cases} z_{ij}, \text{if } i \neq j \\ a_i, \text{if } i = j \end{cases} \tag{5}$$

For the lift operation, we have the following Theorem 3.

**Theorem 3.** *Suppose $T = Q^T D Q$ is an eigen-decomposition of $T$ where $D = diag(\mathbf{d})$ with $\mathbf{d} = [d_1, d_2, ..., d_m]^T$ being $T$'s eigenvalues which are ordered as $d_1 \geq d_2 \geq \cdots \geq d_m$. Then the nearest neighbor of $T$ in $\mathcal{M}(\Lambda)$ is given by*

$$Z = Q^T \Lambda Q. \tag{6}$$

*Proof.* See Theorem 4.1 in [3]. $\qquad\qquad\qquad\qquad\qquad\qquad\qquad\qquad\qquad\qquad\qquad\square$

Since in each step we minimize the distance between $T$ and $Z$, we have

$$||T^{(k)} - Z^{(k)}||_F \geq ||T^{(k+1)} - Z^{(k)}||_F \geq ||T^{(k+1)} - Z^{(k+1)}||_F.$$

It is easy to see that $(T^{(k)}, Z^{(k)})$ will converge to a stationary point. The whole IsoHash algorithm based on LP, abbreviated as IsoHash-LP, is briefly summarized in Algorithm 1.

---

**Algorithm 1** Lift and projection based IsoHash (IsoHash-LP)

---

**Input:** $X \in \mathbb{R}^{d \times n}, m \in \mathbb{N}^+, t \in \mathbb{N}^+$
- $[\Lambda, W] = PCA(X, m)$, as stated in Section 2.2.
- Generate a random orthogonal matrix $Q_0 \in \mathbb{R}^{m \times m}$.
- $Z^{(0)} \leftarrow Q_0^T \Lambda Q_0$.
- **for** $k = 1 \rightarrow t$ **do**
  Calculate $T^{(k)}$ from $Z^{(k-1)}$ by equation (5).
  Perform eigen-decomposition of $T^{(k)}$ to get $Q_k^T D Q_k = T^{(k)}$.
  Calculate $Z^{(k)}$ from $Q_k$ and $\Lambda$ by equation (6).
- **end for**
- $Y = sgn(Q_t^T W^T X)$.

**Output:** $Y$

---

Because $\mathcal{M}(\Lambda)$ is not a convex set, the stationary point we find is not necessarily inside the intersection of $\mathcal{T}(\mathbf{a})$ and $\mathcal{M}(\Lambda)$. For example, if we set $Z^{(0)} = \Lambda$, the lift and projection learning algorithm would get no progress because the $Z$ and $T$ are already in a stationary point. To solve this problem of degenerate solutions, we initiate $Z$ as a transformed $\Lambda$ with some random orthogonal matrix $Q_0$, which is illustrated in Algorithm 1.

### 2.3.2 Gradient Flow

Another learning algorithm is a continuous one based on the construction of a gradient flow (GF) on the surface $\mathcal{M}(\Lambda)$ that moves towards the desired intersection point. Because there always exists a solution for the problem in (3) according to Theorem 2, the objective function in (4) can be reformulated as follows [2]:

$$\min_{Q \in \mathcal{O}(m)} F(Q) = \frac{1}{2} ||\text{diag}(Q^T \Lambda Q) - \text{diag}(\mathbf{a})||_F^2. \tag{7}$$

The details about how to optimize (7) can be found in [2]. We just show some key steps of the learning algorithm in the following content.

The gradient $\nabla F$ at $Q$ can be calculated as

$$\nabla F(Q) = 2\Lambda \beta(Q), \tag{8}$$

where $\beta(Q) = \text{diag}(Q^T \Lambda Q) - \text{diag}(\mathbf{a})$. Once we have computed the gradient of $F$, it can be projected onto the manifold $\mathcal{O}(m)$ according to the following Theorem 4.

**Theorem 4.** *The projection of* $\nabla F(Q)$ *onto* $\mathcal{O}(m)$ *is given by*

$$g(Q) = Q[Q^T \Lambda Q, \beta(Q)] \qquad (9)$$

*where* $[A, B] = AB - BA$ *is the Lie bracket.*

*Proof.* See the formulas (20), (21) and (22) in [3]. $\qquad \qquad \square$

The vector field $\dot{Q} = -g(Q)$ defines a steepest descent flow on the manifold $\mathcal{O}(m)$ for function $F(Q)$. Letting $Z = Q^T \Lambda Q$ and $\alpha(Z) = \beta(Q)$, we get

$$\dot{Z} = [Z, [\alpha(Z), Z]], \qquad (10)$$

where $\dot{Z}$ is an isospectral flow that moves to reduce the objective function $F(Q)$.

As stated by Theorems 3.3 and 3.4 in [2], a stable equilibrium point of (10) must be combined with $\beta(Q) = 0$, which means that $F(Q)$ has decreased to zero. Hence, the gradient flow method can always find an intersection point as the solution. The whole IsoHash algorithm based on GF, abbreviated as IsoHash-GF, is briefly summarized in Algorithm 2.

---

**Algorithm 2** Gradient flow based IsoHash (IsoHash-GF)

**Input:** $X \in \mathbb{R}^{d \times n}, m \in \mathbb{N}^+$

- $[\Lambda, W] = PCA(X, m)$, as stated in Section 2.2.
- Generate a random orthogonal matrix $Q_0 \in \mathbb{R}^{m \times m}$.
- $Z^{(0)} \leftarrow Q_0^T \Lambda Q_0$.
- Start integration from $Z = Z^{(0)}$ with gradient computed from equation (10).
- Stop integration when reaching a stable equilibrium point.
- Perform eigen-decomposition of $Z$ to get $Q^T \Lambda Q = Z$.
- $Y = sgn(Q^T W^T X)$.

**Output:** $Y$

---

We now discuss some implementation details of IsoHash-GF. Since all diagonal matrices in $\mathcal{M}(\Lambda)$ result in $\dot{Z} = \mathbf{0}$, one should not use $\Lambda$ as the starting point. In our implementation, we use the same method as that in IsoHash-LP to avoid this degenerate case, i.e., a random orthogonal transformation matrix $Q_0$ is use to rotate $\Lambda$. To integrate $Z$ with gradient in (10), we use Adams-Bashforth-Moulton PECE solver in [27] where the parameter RelTol is set to $10^{-3}$. The relative error of the algorithm is computed by comparing the diagonal entries of $Z$ to the target $diag(\mathbf{a})$. The whole integration process will be terminated when their relative error is below $10^{-7}$.

### 2.4 Complexity Analysis

The learning of our IsoHash method contains two phases: the first phase is PCA and the second phase is LP or GF. The time complexity of PCA is $O(\min(n^2 d, nd^2))$. The time complexity of LP after PCA is $O(m^3 t)$, and that of GF after PCA is $O(m^3)$. In our experiments, $t$ is set to 100 because good performance can be achieved at this setting. Because $m$ is typically set to be a very small number like 64 or 128, the main time complexity of IsoHash is from the PCA phase. In general, the training of IsoHash-GF will be faster than IsoHash-LP in our experiments.

One promising property of both LP and GF is that the time complexity after PCA is independent of the number of training data, which makes them scalable to large-scale data sets.

## 3 Relation to Existing Works

The most related method of IsoHash is ITQ [7], because both ITQ and IsoHash have to learn an orthogonal matrix. However, IsoHash is different from ITQ in many aspects: firstly, the goal of IsoHash is to learn a projection with isotropic variances, but the results of ITQ cannot necessarily guarantee isotropic variances; secondly, IsoHash directly learns the orthogonal matrix from the eigenvalues and eigenvectors of PCA, but ITQ first quantizes the PCA results to get some binary

codes, and then learns the orthogonal matrix based on the resulting binary codes; thirdly, IsoHash has an explicit objective function to optimize, but ITQ uses a two-step heuristic strategy whose goal cannot be formulated by a single objective function; fourthly, to learn the orthogonal matrix, IsoHash uses Lift and Projection or Gradient Flow, but ITQ uses Procruster method which is much slower than IsoHash. From the experimental results which will be presented in the next section, we can find that IsoHash can achieve accuracy comparable to ITQ with much faster training speed.

## 4 Experiment

### 4.1 Data Sets

We evaluate our methods on two widely used data sets, CIFAR [16] and LabelMe [28].

The first data set is *CIFAR-10* [16] which consists of 60,000 images. These images are manually labeled into 10 classes, which are *airplane*, *automobile*, *bird*, *cat*, *deer*, *dog*, *frog*, *horse*, *ship*, and *truck*. The size of each image is 32×32 pixels. We represent them with 256-dimensional gray-scale GIST descriptors [24].

The second data set is *22K LabelMe* used in [23, 28] which contains 22,019 images sampled from the large LabelMe data set. As in [28], The images are scaled to 32×32 pixels, and then represented by 512-dimensional GIST descriptors [24].

### 4.2 Evaluation Protocols and Baselines

As the protocols widely used in recent papers [7, 23, 25, 31], Euclidean neighbors in the original space are considered as ground truth. More specifically, a threshold of the average distance to the 50th nearest neighbor is used to define whether a point is a true positive or not. Based on the Euclidean ground truth, we compute the precision-recall curve and mean average precision (mAP) [7, 21]. For all experiments, we randomly select 1000 points as queries, and leave the rest as training set to learn the hash functions. All the experimental results are averaged over 10 random training/test partitions.

Although a lot of hashing methods have been proposed, some of them are either supervised [23] or semi-supervised [29]. Our IsoHash method is essentially an *unsupervised* one. Hence, for fair comparison, we select the most representative unsupervised methods for evaluation, which contain PCAH [7], ITQ [7], SH [31], LSH [1], and SIKH [25]. Among these methods, PCAH, ITQ and SH are data-dependent methods, while SIKH and LSH are data-independent methods.

All experiments are conducted on our workstation with Intel(R) Xeon(R) CPU X7560@2.27GHz and 64G memory.

### 4.3 Accuracy

Table 1 shows the Hamming ranking performance measured by mAP on LabelMe and CIFAR. It is clear that our IsoHash methods, including both IsoHash-GF and IsoHash-LP, achieve far better performance than PCAH. The main difference between IsoHash and PCAH is that the PCAH dimensions have anisotropic variances while IsoHash dimensions have isotropic variances. Hence, the intuitive viewpoint that using the same number of bits for different projected dimensions with anisotropic variances is unreasonable has been successfully verified by our experiments. Furthermore, the performance of IsoHash is also comparable, if not superior, to the state-of-the-art methods, such as ITQ.

Figure 1 illustrates the precision-recall curves on LabelMe data set with different code sizes. The relative performance in the precision-recall curves on CIFAR is similar to that on LabelMe. We omit the results on CIFAR due to space limitation. Once again, we can find that our IsoHash methods can achieve performance which is far better than PCAH and comparable to the state-of-the-art.

### 4.4 Computational Cost

Table 2 shows the training time on CIFAR. We can see that our IsoHash methods are much faster than ITQ. The time complexity of ITQ also contains two parts: the first part is PCA which is the same

Table 1: mAP on LabelMe and CIFAR data sets.

| Method | LabelMe | | | | | CIFAR | | | | |
|---|---|---|---|---|---|---|---|---|---|---|
| # bits | 32 | 64 | 96 | 128 | 256 | 32 | 64 | 96 | 128 | 256 |
| IsoHash-GF | 0.2580 | 0.3269 | 0.3528 | 0.3662 | 0.3889 | 0.2249 | 0.2969 | **0.3256** | **0.3357** | 0.3600 |
| IsoHash-LP | 0.2534 | 0.3223 | **0.3577** | **0.3826** | 0.4274 | 0.1907 | 0.2624 | 0.3027 | 0.3223 | **0.3651** |
| PCAH | 0.0516 | 0.0401 | 0.0341 | 0.0307 | 0.0232 | 0.0319 | 0.0274 | 0.0241 | 0.0216 | 0.0168 |
| ITQ | **0.2786** | **0.3328** | 0.3504 | 0.3615 | 0.3728 | **0.2490** | **0.3051** | 0.3238 | 0.3319 | 0.3436 |
| SH | 0.0826 | 0.1034 | 0.1447 | 0.1653 | 0.2080 | 0.0510 | 0.0589 | 0.0802 | 0.1121 | 0.1535 |
| SIKH | 0.0590 | 0.1482 | 0.2074 | 0.2526 | **0.4488** | 0.0353 | 0.0902 | 0.1245 | 0.1909 | 0.3614 |
| LSH | 0.1549 | 0.2574 | 0.3147 | 0.3375 | 0.4034 | 0.1052 | 0.1907 | 0.2396 | 0.2776 | 0.3432 |

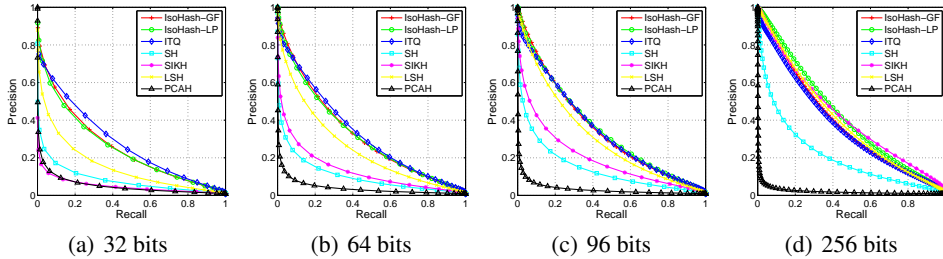

| (a) 32 bits | (b) 64 bits | (c) 96 bits | (d) 256 bits |

Figure 1: Precision-recall curves on LabelMe data set.

as that in IsoHash, and the second part is an iteration algorithm to rotate the original PCA matrix with time complexity $O(nm^2)$, where $n$ is the number of training points and $m$ is the number of bits in the binary code. Hence, as the number of training data increases, the second-part time complexity of ITQ will increase linearly to the number of training points. But the time complexity of IsoHash after PCA is independent of the number of training points. Hence, IsoHash will be much faster than ITQ, particularly in the case with a large number of training points. This is clearly shown in Figure 2 which illustrates the training time when the numbers of training data are changed.

Table 2: Training time (in second) on CIFAR.

| # bits | 32 | 64 | 96 | 128 | 256 |
|---|---|---|---|---|---|
| IsoHash-GF | 2.48 | 2.45 | 2.70 | 3.00 | 5.55 |
| IsoHash-LP | 2.14 | 2.43 | 2.94 | 3.47 | 8.83 |
| PCAH | 1.84 | 2.14 | 2.23 | 2.36 | 2.92 |
| ITQ | 4.35 | 6.33 | 9.73 | 12.40 | 29.25 |
| SH | 1.60 | 3.41 | 8.37 | 13.66 | 49.44 |
| SIKH | 1.30 | 1.44 | 1.57 | 1.55 | 2.20 |
| LSH | 0.05 | 0.08 | 0.11 | 0.19 | 0.31 |

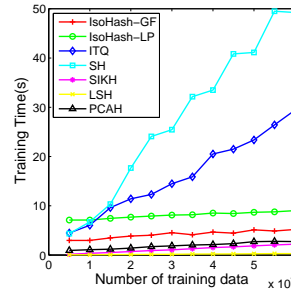

Figure 2: Training time on CIFAR .

## 5 Conclusion

Although many researchers have intuitively argued that using the same number of bits for different projected dimensions with anisotropic variances is unreasonable, this viewpoint has still not been verified by either theory or experiment because no methods have been proposed to find projection functions with isotropic variances for different dimensions. The proposed IsoHash method in this paper is the first work to learn projection functions which can produce projected dimensions with isotropic variances (equal variances). Experimental results on real data sets have successfully verified the viewpoint that projections with isotropic variances will be better than those with anisotropic variances. Furthermore, IsoHash can achieve accuracy comparable to the state-of-the-art methods with faster training speed.

## 6 Acknowledgments

This work is supported by the NSFC (No. 61100125), the 863 Program of China (No. 2011AA01A202, No. 2012AA011003), and the Program for Changjiang Scholars and Innovative Research Team in University of China (IRT1158, PCSIRT).

## Footnotes

[1] Please note in [2], the values are in increasing order. It is easy to prove that our presentation of Schur-Horn lemma is equivalent to that in [2]. The non-increasing order is chosen here just because it will facilitate our following presentation due to the non-increasing order of the eigenvalues in $\Lambda$.

# References

[1] A. Andoni and P. Indyk. Near-optimal hashing algorithms for approximate nearest neighbor in high dimensions. *Commun. ACM*, 51(1):117–122, 2008.

[2] M.T. Chu. Constructing a Hermitian matrix from its diagonal entries and eigenvalues. *SIAM Journal on Matrix Analysis and Applications*, 16(1):207–217, 1995.

[3] M.T. Chu and K.R. Driessel. The projected gradient method for least squares matrix approximations with spectral constraints. *SIAM Journal on Numerical Analysis*, pages 1050–1060, 1990.

[4] M. Datar, N. Immorlica, P. Indyk, and V. S. Mirrokni. Locality-sensitive hashing scheme based on p-stable distributions. In *Proceedings of the ACM Symposium on Computational Geometry*, 2004.

[5] A. Gionis, P. Indyk, and R. Motwani. Similarity search in high dimensions via hashing. In *VLDB*, 1999.

[6] Y. Gong, S. Kumar, V. Verma, and S. Lazebnik. Angular quantization based binary codes for fast similarity search. In *NIPS*, 2012.

[7] Y. Gong and S. Lazebnik. Iterative quantization: A Procrustean approach to learning binary codes. In *CVPR*, 2011.

[8] Y. Gong, S. Lazebnik, A. Gordo, and F. Perronnin. Iterative quantization: A Procrustean approach to learning binary codes for large-scale image retrieval. In *IEEE Trans. Pattern Anal. Mach. Intell.*, 2012.

[9] J. He, W. Liu, and S.-F. Chang. Scalable similarity search with optimized kernel hashing. In *KDD*, 2010.

[10] J.-P. Heo, Y. Lee, J. He, S.-F. Chang, and S.-E. Yoon. Spherical hashing. In *CVPR*, 2012.

[11] A. Horn. Doubly stochastic matrices and the diagonal of a rotation matrix. *American Journal of Mathematics*, 76(3):620–630, 1954.

[12] H. Jegou, M. Douze, C. Schmid, and P. Pérez. Aggregating local descriptors into a compact image representation. In *CVPR*, 2010.

[13] I. Jolliffe. *Principal Component Analysis*. Springer, 2002.

[14] W. Kong and W.-J. Li. Double-bit quantization for hashing. In *AAAI*, 2012.

[15] W. Kong, W.-J. Li, and M. Guo. Manhattan hashing for large-scale image retrieval. In *SIGIR*, 2012.

[16] A. Krizhevsky. Learning multiple layers of features from tiny images. Tech report, University of Toronto, 2009.

[17] B. Kulis and T. Darrell. Learning to hash with binary reconstructive embeddings. In *NIPS*, 2009.

[18] B. Kulis and K. Grauman. Kernelized locality-sensitive hashing for scalable image search. In *ICCV*, 2009.

[19] B. Kulis, P. Jain, and K. Grauman. Fast similarity search for learned metrics. *IEEE Trans. Pattern Anal. Mach. Intell.*, 31(12):2143–2157, 2009.

[20] W. Liu, J. Wang, R. Ji, Y.-G. Jiang, and S.-F. Chang. Supervised hashing with kernels. In *CVPR*, 2012.

[21] W. Liu, J. Wang, S. Kumar, and S.-F. Chang. Hashing with graphs. In *ICML*, 2011.

[22] Y. Mu and S. Yan. Non-metric locality-sensitive hashing. In *AAAI*, 2010.

[23] M. Norouzi and D. J. Fleet. Minimal loss hashing for compact binary codes. In *ICML*, 2011.

[24] A. Oliva and A. Torralba. Modeling the shape of the scene: A holistic representation of the spatial envelope. *International Journal of Computer Vision*, 42(3):145–175, 2001.

[25] M. Raginsky and S. Lazebnik. Locality-sensitive binary codes from shift-invariant kernels. In *NIPS*, 2009.

[26] R. Salakhutdinov and G. E. Hinton. Semantic hashing. *Int. J. Approx. Reasoning*, 50(7):969–978, 2009.

[27] L.F. Shampine and M.K. Gordon. Computer solution of ordinary differential equations: the initial value problem. *Freeman, San Francisco, California*, 1975.

[28] A. Torralba, R. Fergus, and Y. Weiss. Small codes and large image databases for recognition. In *CVPR*, 2008.

[29] J. Wang, S. Kumar, and S.-F. Chang. Sequential projection learning for hashing with compact codes. In *ICML*, 2010.

[30] J. Wang, S. Kumar, and S.-F. Chang. Semi-supervised hashing for large-scale search. *IEEE Trans. Pattern Anal. Mach. Intell.*, 34(12):2393–2406, 2012.

[31] Y. Weiss, A. Torralba, and R. Fergus. Spectral hashing. In *NIPS*, 2008.

[32] H. Xu, J. Wang, Z. Li, G. Zeng, S. Li, and N. Yu. Complementary hashing for approximate nearest neighbor search. In *ICCV*, 2011.

[33] D. Zhang, F. Wang, and L. Si. Composite hashing with multiple information sources. In *SIGIR*, 2011.

[34] Y. Zhen and D.-Y. Yeung. A probabilistic model for multimodal hash function learning. In *KDD*, 2012.

